# Structural epitome: A way to summarize one's visual experience

**Nebojsa Jojic**
Microsoft Research

**Alessandro Perina**
Microsoft Research
University of Verona

**Vittorio Murino**
Italian Institute of Technology
University of Verona

## Abstract

In order to study the properties of total visual input in humans, a single subject wore a camera for two weeks capturing, on average, an image every 20 seconds. The resulting new dataset contains a mix of indoor and outdoor scenes as well as numerous foreground objects. Our first goal is to create a visual summary of the subject's two weeks of life using unsupervised algorithms that would automatically discover recurrent scenes, familiar faces or common actions. Direct application of existing algorithms, such as panoramic stitching (e.g., Photosynth) or appearance-based clustering models (e.g., the epitome), is impractical due to either the large dataset size or the dramatic variations in the lighting conditions. As a remedy to these problems, we introduce a novel image representation, the "structural element (stel) epitome," and an associated efficient learning algorithm. In our model, each image or image patch is characterized by a hidden mapping T which, as in previous epitome models, defines a mapping between the image coordinates and the coordinates in the large "all-I-have-seen" epitome matrix. The limited epitome real-estate forces the mappings of different images to overlap which indicates image similarity. However, the image similarity no longer depends on direct pixel-to-pixel intensity/color/feature comparisons as in previous epitome models, but on spatial configuration of scene or object parts, as the model is based on the palette-invariant stel models. As a result, stel epitomes capture structure that is invariant to non-structural changes, such as illumination changes, that tend to uniformly affect pixels belonging to a single scene or object part.

## 1 Introduction

We develop a novel generative model which combines the powerful invariance properties achieved through the use of hidden variables in epitome [2] and stel (structural element) models [6, 8]. The latter set of models have a hidden stel index $s_i$ for each image pixel $i$. The number of discrete states $s_i$ can take is small, typically 4-10, as the stel indices point to a small palette of distributions over local measurements, e.g., color. The actual local measurement $x_i$ (e.g. color) for pixel $i$ is assumed to have been generated from the appropriate palette entry. This constrains the pixels with the same stel index $s$ to have similar colors or whatever local measurements $x_i$ represent. The indexing scheme is further assumed to change little accross different images of the same scene/object, while the palettes can vary significantly. For example, two images of the same scene captured in different levels of overall illumination would still have very similar stel partitions, even though their palettes may be vastly different. In this way, the image representation rises above a matrix of local measurements in favor of a matrix of stel indices which can survive remarkable non-structural image changes, as long as these can be explained away by a change in the (small) palette. For example, in Fig. 1B, images of pedestrians are captured by a model that has a prior distribution of stel assignments shown in the first row. The prior on stel probabilities for each pixel adds up to one, and the 6 images showing these prior probabilities add up to a uniform image of ones. Several pedestrian images are shown

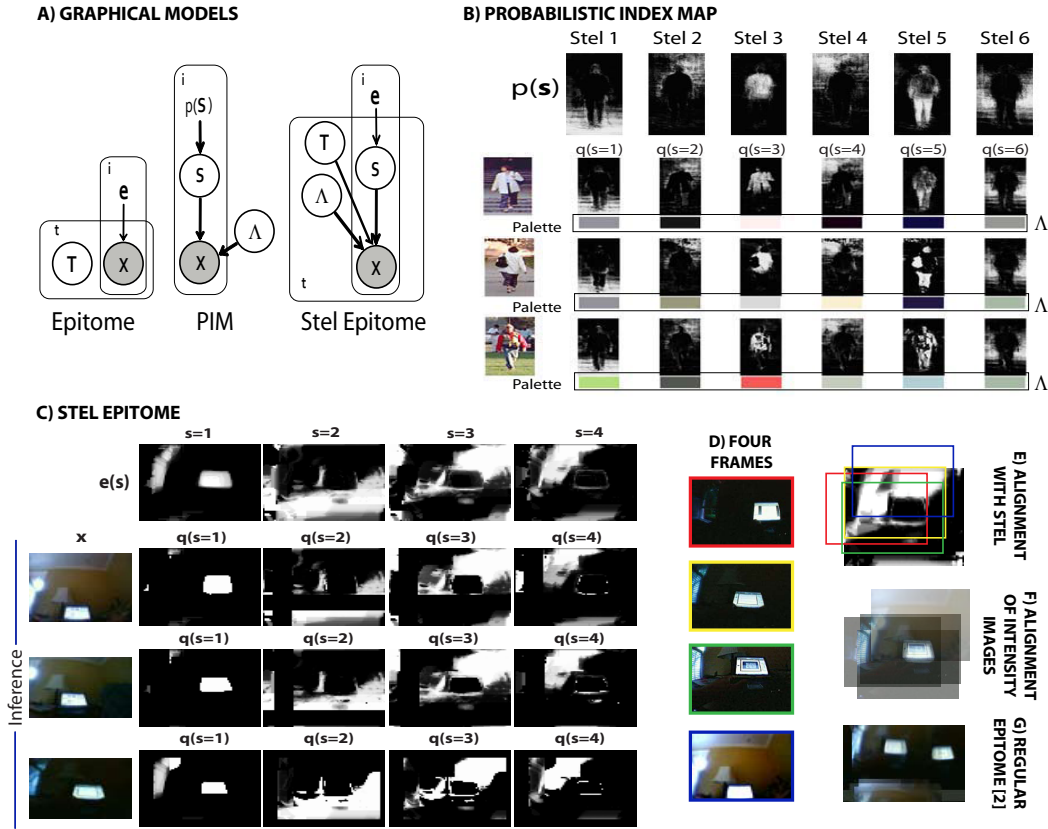

Figure 1: A) Graphical model of Epitome, Probabilistic index map (PIM) and Stel epitome. B) Examples of PIM parameters. C) Example of stel epitome parameters. D) Four frames aligned with stel epitome E-F). In G) we show the original epitome model [2] trained on these four frames.

below with their *posterior* distributions over stel assignments, as well as the mean color of each stel. This illustrates that the different parts of the pedestrian images are roughly matched. Torso pixels, for instance, are consistently assigned to stel $s = 3$, despite the fact that different people wore shirts or coats of very different colors. Such a consistent segmentation is possible because torso pixels tend to have similar colors within *any* given image and because the torso is roughly in the same position across images (though misalignment of up to half the size of the segments is largely tolerated). While the figure shows the model with S=6 stels, larger number of stels were shown to lead to further segmentation of the head and even splitting of the left from right leg [6]. Motivated by similar insights as in [6], a number of models followed, e.g. [7, 13, 14, 8], as the described addition of hidden variables $s$ achieves the remarkable level of intensity invariance first demonstrated through the use of similarity templates [12], but at a much lower computational cost.

In this paper, we embed the stel image representation within a large stel epitome: a stel prior matrix, like the one shown in the top row of Fig. 1B, but much larger so that it can contain representations of multiple objects or scenes. This requires the additional transformation variables $T$ for each image whose role is to align it with the epitome. The model is thus qualitatively enriched in two ways: 1) the model is now less sensitive to misalignment of images, as through alignment to the epitome, the images are aligned to each other, and 2) interesting structure emerges when the epitome real estate is limited so that though it is much larger than the size of a single image, it is till much smaller than the real estate needed to simply tile all images without overlap. In that case, a large collection of images must naturally undergo an unsupervised clustering in order for this real estate to be used as well as possible (or as well as the local minimum obtained by the learning algorithm allows). This clustering is quite different from the traditional notion of clustering. As in the original epitome models, the transformation variables play both the alignment and cluster indexing roles. Different

models over the typical scenes/objects have to compete over the positions in the epitome, with a panoramic version of each scene emerging in different parts of the epitome, finally providing a rich image indexing scheme. Such a panoramic scene submodel within the stel epitome is illustrated in Fig. 1C. A portion of the larger stel epitome is shown with 3 images that map into this region. The region represents one of two home offices in the dataset analyzed in the experiments. Stel s=1 captures the laptop screen, while the other stels capture other parts of the scene, as well as large shadowing effects (while the overall changes in illumination and color changes in object parts rarely affect stel representations, the shadows can break the stel invariance, and so the model learned to cope with them by breaking the shadows across multiple stels). The three images shown, mapping to different parts of this region, have very different colors as they were taken at different times of day and across different days, and yet their alignment is not adversely affected, as it is evident in their posterior stel segmentation aligned to the epitome.

To further illustrate the panoramic alignment, we used the epitome mapping to show for the 4 different images in Fig. 1D how they overlap with stel s=4 of another office image (Fig. 1E), as well as how multiple images of this scene, including these 4, look when they are aligned and overlapped as intensity images in Fig. 1F. To illustrate the gain from palette-invariance that motivated this work, we show in Fig. 1G the original epitome model [2] trained on images of this scene. Without the invariances afforded by the stel representation, the standard color epitome has to split the images of the scene into two clusters, and so the laptop screen is doubled there.

Qualitatively quite different from both epitomes and previous stel models, the stel epitome is a model flexible enough to be applied to a very diverse set of images. In particular, we are interested in datasets that might represent well a human's *total* visual input over a longer period of time, and so we captured two weeks worth of SenseCam images, taken at a frequency of roughly one image every 20 seconds during all waking hours of a human subject over a period of two weeks (www.research.microsoft.com/~jojic/aihs).

## 2 Stel epitome

The graphical model describing the dependencies in stel epitomes is provided in Fig. 1A. The parametric forms for the conditional distributions are standard multinomial and Gaussian distributions just as the ones used in [8]. We first consider the generation of a single image or an image patch (depending on which visual scale we are epitomizing), and, for brevity, temporarily omit the subscript $t$ indexing different images.

The epitome is a matrix of multinomial distributions over $S$ indices $s \in \{1, 2, ..., S\}$, associated with each two-dimensional epitome location $i$:

$$p(s_i = s) = e_i(s). \tag{1}$$

Thus each location in the epitome contains $S$ probabilities (adding to one) for different indices. Indices for the image are assumed to be generated from these distributions. The distribution over the entire collection of pixels (either from an entire image, or a patch), $p(\{x_i\}|\{s_i\}, T, \Lambda)$, depends on the parametrization of the transformations $T$. We adopt the discrete transformation model used previously in graphical models e.g. [1, 2], where the shifts are separated from other transformations such as scaling or rotation, $T = (\ell, r)$, with $\ell$ being a 2-dimensional shift and $r$ being the index into the set of other transformations, e.g., combinations of rotation and scaling:

$$p(\{x_i\}|\{s_i\}, T, \Lambda) = \prod_i p(x_{i-\ell}^r|s_i, \Lambda) = \prod_i p(x_{i-\ell}^r|\Lambda_{s_i}), \tag{2}$$

where superscript $r$ indicates transformation of the image $x$ by the $r$-th transformation, and $i - \ell$ is the mod-difference between the two-dimensional variables with respect to the edges of the epitome (the shifts wrap around). $\Lambda$ is the palette associated with the image, and $\Lambda_s$ is its $s - th$ entry. Various palette models for probabilistic index / structure element map models have been reviewed in [8]. For brevity, in this paper we focus on the simplest case where the image measurements are simply pixel colors, and the palette entries are simply Gaussians with parameters $\Lambda_s = (\mu_s, \phi_s)$. In this case, $p(x_{i-\ell}^r|\Lambda_{s_i}) = \mathcal{N}(x_{i-\ell}^r; \mu_{s_i}, \phi_{s_i})$, and the joint likelihood over observed and hidden variables can be written as

$$P = p(\Lambda)p(\ell, r) \prod_i \prod_s \left( \mathcal{N}(x_{i-\ell}^r; \mu_s, \phi_s) e_i(s) \right)^{[s_i=s]}, \tag{3}$$

where [] is the indicator function.

To derive the inference and leaning algorithms for the mode, we start with a posterior distribution model $Q$ and the appropriate free energy $\sum Q \log \frac{Q}{P}$. The standard variational approach, however, is not as straightforward as we might hope as major obstacles need to be overcome to avoid local minima and slow convergence. To focus on these important issues, we further simplify the problem and omit both the non-shift part of the transformations ($r$) and palette priors $p(\Lambda)$, and for consistency, we also omit these parts of the model in the experiments. These two elements of the model can be dealt with in the manner proposed previously: The $R$ discrete transformations (scale/rotation combinations, for example) can be inferred in a straight-forward way that makes the entire algorithm that follows $R$ times slower (see [1] for using such transformations in a different context), and the various palette models from [8] can all be inserted here with the update rules adjusted appropriately.

A large stel epitome is difficult to learn because decoupling of all hidden variables in the posterior leads to severe local minima, with all images either mapped to a single spot in the epitome, or mapped everywhere in the epitome so that the stel distribution is flat. This problem becomes particularly evident in larger epitomes, due to the imbalance in the cardinalities of the three types of hidden variables. To resolve this, we either need a very high numerical precision (and considerable patience), or the severe variational approximations need to be avoided as much as possible. It is indeed possible to tractably use a rather expressive posterior

$$Q = q(\ell) \prod_s q(\Lambda_s | \ell) \prod_i q(s_i), \tag{4}$$

further setting $q(\Lambda_s | \ell) = \delta(\mu_s - \hat{\mu}_{s,\ell}) \delta(\phi_s - \hat{\phi}_{s,\ell})$, where $\delta$ is the Dirac function. This leads to

$$
\begin{aligned}
F &= H(Q) + \sum_{s,\ell,i} q(\ell) q(s_i = s) \frac{x_{i-\ell}^2}{2\hat{\phi}_{s,\ell}} - \sum_{s,\ell,i} q(\ell) q(s_i = s) \frac{\hat{\mu}_{s,\ell} x_{i-\ell}}{\hat{\phi}_{s,\ell}} + \\
&\quad + \sum_{s,\ell,i} q(\ell) q(s_i = s) \frac{\hat{\mu}_{s,\ell}^2}{2\hat{\phi}_{s,\ell}} - \sum_s \sum_i q(s_i = s) \log e_i(s),
\end{aligned} \tag{5}
$$

where $H(Q)$ is the entropy of the posterior distribution. Setting to zero the derivatives of this free energy with respect to the variational parameters – the probabilities $q(s_i = s)$, $q(\ell)$, and the palette means and variance estimates $\hat{\mu}_{s,\ell}, \hat{\phi}_{s,\ell}$ – we obtain a set of updates for iterative inference.

## 2.1  E STEP

The following steps are iterated for a single image $x$ on an $m \times n$ grid and for a given epitome distributions $e(s)$ on an $M \times N$ grid. Index $i$ corresponds to the epitome coordinates and masks $m$ are used to describe which of all $M \times N$ coordinates correspond to image coordinates. In the variational EM learning on a collection of images index by $t$, these steps are done for each image, yielding posterior distributions indexed by $t$ and then the M step is performed as described below.

We initialize $q(s_i = s) = e(s_i)$ and then iterate the following steps in the following order.

**Palette updates**

$$\hat{\mu}_{s,\ell} = \frac{\sum_\ell \sum_i m_{i-\ell} q(s_i = s) q(\ell) x_{i-\ell}}{\sum_\ell \sum_i q(s_i = s) q(\ell) m_{i-\ell}} \tag{6}$$

$$\hat{\phi}_{s,\ell} = \left( \frac{\sum_\ell \sum_i m_{i-\ell} q(s_i = s) q(\ell) x_{i-\ell}^2}{\sum_\ell \sum_i q(s_i = s) q(\ell) m_{i-\ell}} \right) - \hat{\mu}_{s,\ell}^2 \tag{7}$$

**Epitome mapping update**

$$\log q(\ell) = const + \frac{1}{2} \sum_{i,s} q(s_i^t = s) \log 2\pi \phi_{i-\ell} \tag{8}$$

This update is derived from the free energy and from the expression for $\phi$ above). This equation can be used as is when the epitome $e(s)$ is well defined (that is the entropy of component stel

distribution is low in the latter iterations), as long as the usual care is taken in exponentiation before normalization - the maximum $\log q(\ell)$ should be subtracted from all elements of the $M \times N$ matrix $\log q(\ell)$ before exponentiation.

In the early iterations of EM, however, when distributions $e_i(s)$ have not converged yet, numerical imprecision can stop the convergence, leaving the algorithm at a point which is not even a local minimum. The reason for this is that after the normalization step we described, $q(\ell)$ will still be very peaky, even for relatively flat $e(s)$ due to the large number of pixels in the image. The consequence is that low alignment probabilities are rounded down to zero, as after exponentiation and normalization their values go below numerical precision. If there are areas of the epitome where no single image is mapped with high probability, then the update in those areas in the M step would have to depend on the low-probability mappings for different images, and their *relative* probabilities would determine which of the images contribute more and which less to updating these areas of the epitome. To preserve the numerical precision needed for this, we set k thresholds $\tau_k$, and compute $\log \tilde{q}(\ell)_k$, the distributions at the $k$ different precision levels:

$$\log \tilde{q}(\ell)_k = [\log q(\ell) \geq \tau_k] \cdot \tau_k + [\log q(\ell) < \tau_k] \cdot \log q(\ell),$$

where [] is the indicator function. This limits how high the highest probability in the map is allowed to be. The $k - th$ distribution sets all values above $\tau_k$ to be equal to $\tau_k$.

We can now normalize these k distributions as discussed above:

$$\tilde{q}(\ell)_k = \frac{\exp\{\log \tilde{q}(\ell)_k - \max_i \log \tilde{q}(\ell)_k\}}{\sum_\ell \exp\{\log \tilde{q}(\ell)_k - \max_i \log \tilde{q}(\ell)_k\}}$$

To keep track of which precision level is needed for different $\ell$, we calculate the masks

$$\tilde{m}_{i,k} = \sum_\ell \tilde{q}(\ell)_k \cdot m_{i-\ell},$$

where mask $m$ is the mask discussed in the main text with ones in the upper left corner's $m \times n$ entries and zeros elsewhere, designating the default image position for a shift of $\ell = 0$ (or given that shifts are defined with a wrap-around, the shift of $\ell = (M, N)$). Masks $\tilde{m}_{i,k}$ provide total weight of the image mapping at the appropriate epitome location at different precision levels.

**Posterior stel distribution q(s) update at multiple precision levels**

$$\log \tilde{q}(s_i = s)_k = \quad const - \sum_\ell \sum_{i|i-\ell \in \mathcal{C}} \tilde{q}(\ell)_k \frac{x_{i-\ell}^2}{2\hat{\phi}_{s,\ell}} + \sum_\ell \sum_{i|i-\ell \in \mathcal{C}} \tilde{q}(\ell)_k \frac{\hat{\mu}_{s,\ell} x_{i-\ell}}{\hat{\phi}_{s,\ell}} -$$

$$- \sum_\ell \sum_{i|i-\ell \in \mathcal{C}} \tilde{q}(\ell)_k \frac{\hat{\mu}_{s,\ell}^2}{2\hat{\phi}_{s,\ell}} + \tilde{m}_{i,k} \cdot \log e(s_i = s). \tag{9}$$

To keep track of these different precision levels, we also define a mask $M$ so that $M_i = k$ indicates that the $k$-th level of detail should be used for epitome location $i$. The $k$-th level is reserved for those locations that have only the values from up to the $k$-th precision band of $q(\ell)$ mapped there (we will have $m \times n$ mappings of the original image to each epitome location, as this many different shifts will align the image so as to overlap with any given epitome location). One simple, though not most efficient way to define this matrix is $M_i = 1 + \lfloor \sum_k \tilde{m}_{i,k} \rfloor$.

We now normalize $\log \tilde{q}(s_i = s)_k$ to compute the distribution at $k$ different precision levels, $\tilde{q}(s_i = s)_k$, and compute $q(s)$ integrating the results from different numerical precision levels as $q(s_i = s) = \sum_k [M_i = k] \cdot \tilde{q}(s_i = s)_k$.

## 2.2 M STEP

The highest $k$ for each epitome location $D_i = \max_t \{M_i^t\}$, is determined over all images $x^t$ in the dataset, so that we know the appropriate precision level at which to perform summation and normalization. Then the epitome update consists of:

$$e(s_i = s) = \sum_k [D_i = k] \frac{\sum_t [M^t = k] \cdot q^t(s_i)}{\sum_t [M^t = k]}.$$

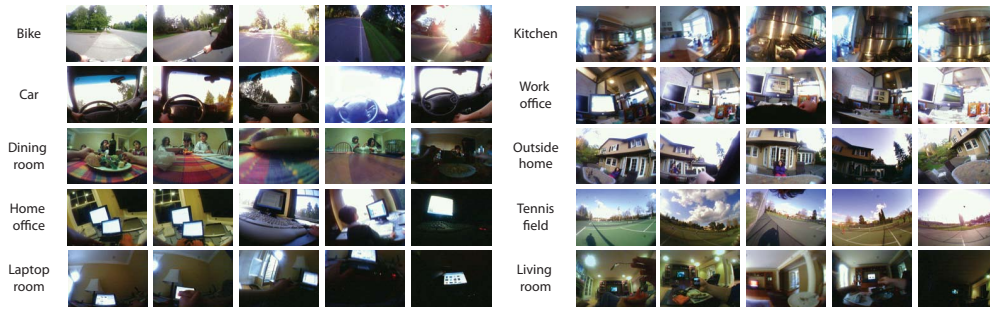

Figure 2: Some examples from the dataset (www.resaerch.microsoft.com/~jojic/aihs)

Note that most of the summations can be performed by convolution operations and as the result, the complexity of the algorithm is of the $O(SMN \log MN)$ for M X N epitomes.

## 3   Experiments

Using a SenseCam wearable camera, we have obtained two weeks worth of images, taken at the rate of one frame every 20 seconds during all waking hours of a human subject. The resulting image dataset captures the subject's (summer) life rather completely in the following sense: Majority of images can be assigned to one of the emergent categories (Fig. 2) and the same categories represent the majority of images from any time period of a couple of days. We are interested in appropriate summarization, browsing, and recognition tasks on this dataset. This dataset also proved to be fundamental for testing stel epitomes, as the illumination and viewing angle variations are significant across images and we found that the previous approaches to scene recognition provide only modest recognition rates. For the purposes of evaluation, we have manually labeled a random collection of 320 images and compared our method with other approaches on supervised and unsupervised classification. We divided this reduced dataset in 10 different recurrent scenes (32 images per class); some examples are depicted in Fig. 2. In all the experiments with the reduced dataset we used an epitome area 14 times larger than the image area and five stels (S=5). The numerical results reported in the tables are averaged over 4 train/test splits.

In supervised learning the scene labels are available during the stel epitome learning. We used this information to aid both the original epitome [9] and the stel epitome modifying the models by the addition of an observed scene class variable $c$ in two ways: i) by linking $c$ in the Bayesian network with $e$, and so learning $p(e|c)$, and ii) by linking $c$ with $T$ inferring $p(T|c)$. In the latter strategy, where we model $p(T|c)$, we learn a single epitome, but we assume that the epitome locations are linked with certain scenes, and this mapping is learned for each epitome pixel. Then, the distribution $p(c|\ell)$ over scene labels can be used for inference of the scene label for the test data. For a previously unseen test image $x^t$, recognition is achieved by computing the label posterior $p(c^t|x^t)$ using $p(c^t|x^t) = \sum_\ell p(c|\ell) \cdot p(\ell|x^t)$.

We compared our approach with the epitomic location recognition method presented in [9], with Latent Dirichlet allocation (LDA) [4], and with the Torralba approach [11]. We also compared with baseline discriminative classifiers and with the pyramid matching kernel approach [5], using SIFT features [3]. For the above techniques that are based on topic models, representing images as spatially disorganized bags of features, the codebook of SIFT features was based 16x16 pixel patches computed over a grid spaced by 8 pixels. We chose a number of topics Z = 45 and 200 codewords (W = 200). The same quantized dictionary has been employed in [5].

To provide a fair comparison between generative and discriminative methods, we also used the free energy optimization strategy presented in [10], which provides an extra layer of discriminative training for an arbitrary generative model. The comparisons are provided in Table 1. Accuracies achieved using the free energy optimization strategy [10] are reported in the Opt. column.

Table 1: Classification accuracies.

| Method | | Accuracy | [10] Opt. | Method | | Accuracy | [10] Opt. |
|---|---|---|---|---|---|---|---|
| Stel epitome | $p(T\|c)$ | 70,06% | n.a. | LDA [4] | | 74,23% | 80,11% |
| Stel epitome | $p(\mathbf{e}\|c)$ | 88,67% | 98,70% | GMM [11] | C=3 | 56,81% | n.a. |
| Epitome [9] | $p(T\|c)$ | 74,36% | n.a. | SIFT + K-NN | | 79,42% | n.a. |
| Epitome [9] | $p(\mathbf{e}\|c)$ | 69,80% | 79,14% | [5] | | 96,67% | n.a. |

We also trained both the regular epitome and the stel epitome in an unsupervised way. An illustration of the resulting stel epitome is provided in Fig. 3. The 5 panels marked $s = 1, \ldots, 5$ show the stel epitome distribution. Each of these panels is an image $e_i(s)$ for an appropriate s. On the top of the stel epitome, four enlarged epitome regions are shown to highlight panoramic reconstructions of a few classes. We also show the result of averaging all images according to their mapping to the stel epitome (Fig. 3D) for comparison with the traditional epitome (Fig.3C) which models colors rather than stels. As opposed to the stel epitome, the learned color epitome [2] has to have multiple versions of the same scene in different illumination conditions. Furthermore, many different scenes tend to overlap in the color epitome, especially indoor scenes which all look equally beige. Finally, in Fig. 3B we show examples of some images of different scenes mapped onto the stel epitome, whose organization is illustrated by a rendering of all images averaged into the appropriate location (similarly to the original color epitomes). Note that the model automatically clusters images using the structure, and not colors, even in face of variation of colors present in the exemplars of the "Car", or the "Work office" classes (See also the supplemental video that illustrates the mapping dynamically). The regular epitome cannot capture these invariances, and it clusters images based on overall intensity more readily than based on the structure of the scene. We evaluated the two models numerically in the following way. Using the two types of unsupervised epitomes, and the known labels for the images in the training set, we assigned labels to the test set using the same classification rule explained in the previous paragraph. This semi-supervised test reveals how consistent the clustering induced by epitomes is with the human labeling. The stel epitome accuracy, 73,06%, outperforms the standard epitome model [9], 69,42%, with statistical significance.

We have also trained both types of epitomes over a real estate 35 times larger than the original image size using different random sets of 5000 images taken from the dataset. The stel epitomes trained in an unsupervised way are qualitatively equivalent, in that they consistently capture around six of the most prominent scenes from Fig. 2, whereas the traditional epitomes tended to capture only three.

## 4    Conclusions

The idea of recording our experiences is not new. (For a review and interesting research directions see [15]). It is our opinion that recording, summarizing and browsing continuous visual input is particularly interesting. With the recent substantial increases in radio connectiviy, battery life, display size, and computing power of small devices, and the avilability of even greater computing power off line, summarizing one's total visual input is now both a practically feasible and scientifically interesting target for vision research. In addition, a variety of applications may arise once this functionality is provided. As a step in this direction, we provide a new dataset that contains a mix of indoor and outdoor scenes as a result of two weeks of continuous image acquisition, as well as a simple algorithm that deals with some of the invariances that have to be incorporated in a model of such data. However, it is likely that modeling the geometry of the imaging process will lead to even more interesting results. Although straightforward application of panoramic stitching algorithms, such as Photosynth, did not work on this dataset, because of both the sheer number of images and the significant variations in the lighting conditions, such methods or insights from their development will most likely be very helpful in further development of unsupervised learning algorithms for such types of datasets. The geometry constraints may lead to more reliable background alignments for the next logical phase in modeling for "All-I-have-seen" datasets: The learning of the foreground object categories such as family members' faces. As this and other such datasets grow in size, the unsupervised techniques for modeling the data in a way where interesting visual components emerge over time will become both more practically useful and scientifically interesting.

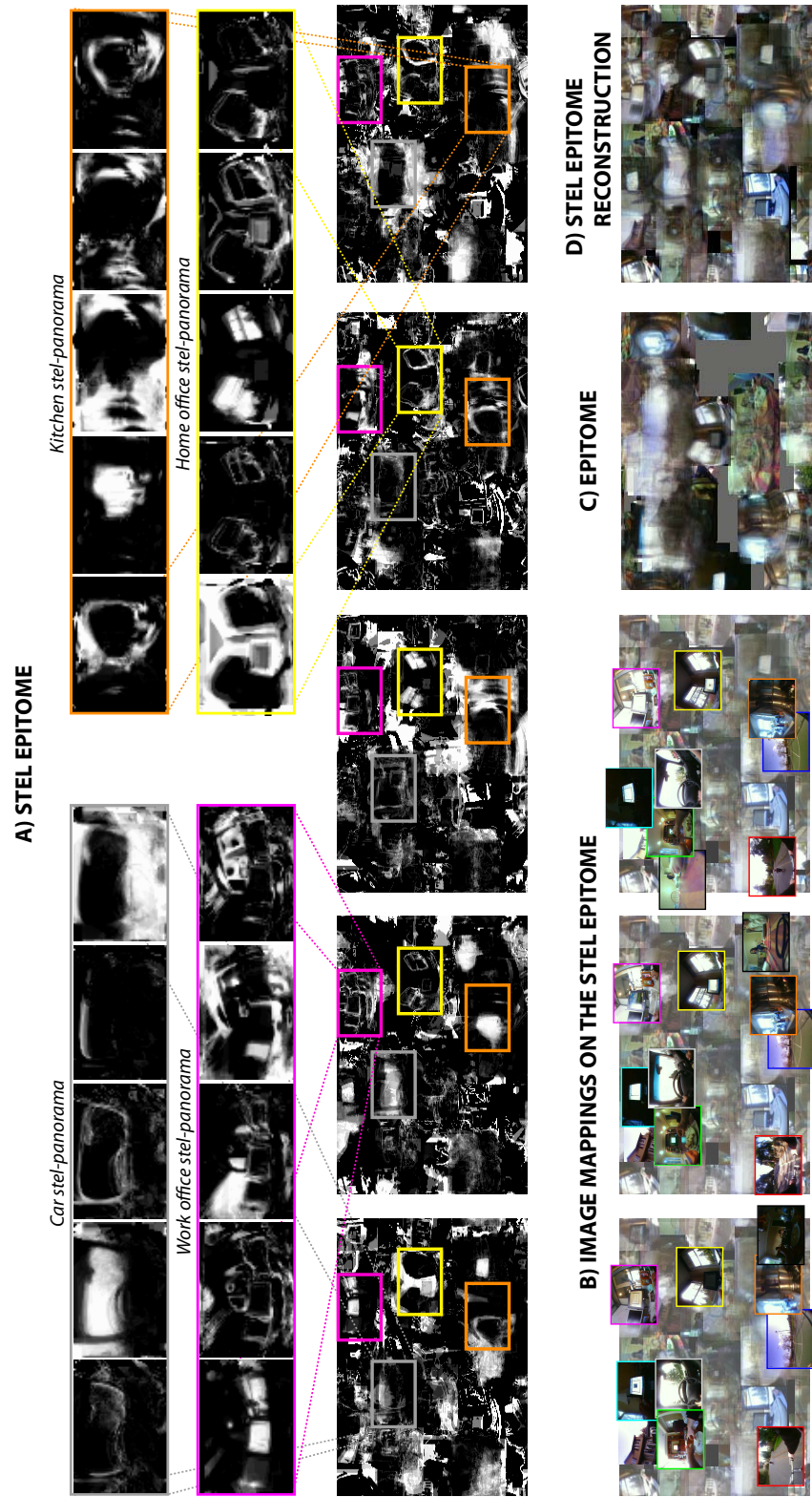

Figure 3: Stel epitome of images captured by a wearable camera

# References

[1] B. Frey and N. Jojic, "Transformation-invariant clustering using the EM algorithm ", TPAMI 2003, vol. 25, no. 1, pp. 1-17.

[2] N. Jojic, B. Frey, A. Kannan, "Epitomic analysis of appearance and shape", ICCV 2003.

[3] D. Lowe, "Distinctive Image Features from Scale-Invariant Keypoints," IJCV, 2004, vol. 60, no. 2, pp. 91-110.

[4] L. Fei-Fei, P. Perona, "A Bayesian Hierarchical Model for Learning Natural Scene Categories," IEEE CVPR 2005, pp. 524-531.

[5] S. Lazebnik, C. Schmid, J. Ponce, "Beyond Bags of Features: Spatial Pyramid Matching for Recognizing Natural Scene Categories," IEEE CVPR, 2006, pp. 2169-2178.

[6] N. Jojic and C. Caspi, "Capturing image structure with probabilistic index maps," IEEE CVPR 2004, pp. 212-219.

[7] J. Winn and N. Jojic, "LOCUS: Learning Object Classes with Unsupervised Segmentation" ICCV 2005.

[8] N. Jojic, A.Perina, M.Cristani, V.Murino and B. Frey, "Stel component analysis: modeling spatial correlation in image class structure," IEEE CVPR 2009.

[9] K. Ni, A. Kannan, A. Criminisi and J. Winn, "Epitomic Location Recognition," IEEE CVPR 2008.

[10] A. Perina, M. Cristani, U. Castellani, V. Murino and N. Jojic, "Free energy score-space," NIPS 2009.

[11] A. Torralba, K.P. Murphy, W.T. Freeman and M.A. Rubin, "Context-based vision system for place and object recognition," ICCV 2003, pp. 273-280.

[12] C. Stauffer, E. Miller, and K. Tieu, "Transform invariant image decomposition with similarity templates," NIPS 2003.

[13] V. Ferrari , A. Zisserman, "Learning Visual Attributes," NIPS 2007.

[14] B. Russell, A. Efros, J. Sivic, B. Freeman, A. Zisserman "Segmenting Scenes by Matching Image Composites," NIPS 2009.

[15] G. Bell and J. Gemmell, Total Recall. Dutton Adult 2009.

